# Universal Consistency of Multi-Class Support Vector Classification

**Tobias Glasmachers**
Dalle Molle Institute for Artificial Intelligence (IDSIA), 6928 Manno-Lugano, Switzerland
`tobias@idsia.ch`

## Abstract

Steinwart was the first to prove universal consistency of support vector machine classification. His proof analyzed the 'standard' support vector machine classifier, which is restricted to binary classification problems. In contrast, recent analysis has resulted in the common belief that several extensions of SVM classification to more than two classes are inconsistent.
Countering this belief, we prove the universal consistency of the multi-class support vector machine by Crammer and Singer. Our proof extends Steinwart's techniques to the multi-class case.

Erratum, 20.01.2011

Unfortunately this paper contains a subtle flaw in the proof of Lemma 5. Furthermore it turns out the statement itself is wrong: The multi-class SVM by Crammer&Singer is *not* universally consistent.

## 1   Introduction

Support vector machines (SVMs) as proposed in [1, 8] are powerful classifiers, especially in the binary case of two possible classes. They can be extended to multi-class problems, that is, problems involving more than two classes, in multiple ways which all reduce to the standard machine in the binary case.

This is trivially the case for general techniques such as one-versus-one architectures and the one-versus-all approach, which combine a set of binary machines to a multi-class decision maker. At least three different 'true' multi-class SVM extensions have been proposed in the literature: The canonical multi-class machine proposed by Vapnik [8] and independently by Weston and Watkins [9], the variant by Crammer and Singer [2], and a conceptually different extension by Lee, Lin, and Wahba [4].

Recently, consistency of multi-class support vector machines has been investigated based on properties of the loss function $\Psi$ measuring empirical risk in machine training [7]. The analysis is based on the technical property of *classification calibration* (refer to [7] for details). This work is conceptually related to Fisher consistency, in contrast to univeral statistical consistency, see [3, 5]. Schematically, Theorem 2 by Tewari and Bartlett [7] establishes the relation

$$\mathcal{S}_A \Leftrightarrow (\mathcal{S}_B \Rightarrow \mathcal{S}_C) \ , \tag{1}$$

for the terms

$\mathcal{S}_A$: The loss function $\Psi$ is classification calibrated.

$\mathcal{S}_B$: The $\Psi$-risk of a sequence $(\hat{f}_n)_{n \in \mathbb{N}}$ of classifiers converges to the minimal possible $\Psi$-risk: $\lim_{n \to \infty} \mathcal{R}_\Psi(\hat{f}_n) = \mathcal{R}_\Psi^*$.

$\mathcal{S}_C$: The 0-1-risk of the same sequence $(\hat{f}_n)_{n\in\mathbb{N}}$ of classifiers converges to the minimal possible 0-1-risk (Bayes risk): $\lim_{n\to\infty} \mathcal{R}(\hat{f}_n) = \mathcal{R}^*$.

The classifiers $\hat{f}_n$ are assumed to result from structural risk minimization [8], that is, the space $F_n$ for which we obtain $\hat{f}_n = \arg\min\{\mathcal{R}_\Psi(f) \mid f \in F_n\}$ grows suitably with the size of the training set such that $\mathcal{S}_B$ holds.

The confusion around the consistency of multi-class machines arises from mixing the equivalence and the implication in statement (1). Examples 1 and 2 in [7] show that the loss functions $\Psi$ used in the machines by Crammer and Singer [2] and by Weston and Watkins [9] are not classification calibrated, thus $\mathcal{S}_A = \mathrm{false}$. Then it is deduced that the corresponding machines are not consistent ($\mathcal{S}_C = \mathrm{false}$), although it can be deduced only that the implication $\mathcal{S}_B \Rightarrow \mathcal{S}_C$ does not hold. This tells us nothing about $\mathcal{S}_C$, even if $\mathcal{S}_B$ can be established per construction.

We argue that the consistency of a machine is not necessarily determined by properties of its loss function. This is because for SVMs it is necessary to provide a sequence of regularization parameters in order to make the infinite sample limit well-defined. Thus, we generalize Steinwart's universal consistency theorem for binary SVMs (Theorem 2 in [6]) to the multi-class support vector machine [2] proposed by Crammer and Singer:

**Theorem 2.** Let $X \subset \mathbb{R}^d$ be compact and $k : X \times X \to \mathbb{R}$ be a universal kernel with[1] $\mathcal{N}((X, d_k), \varepsilon) \in \mathcal{O}(\varepsilon^{-\alpha})$ for some $\alpha > 0$. Suppose that we have a positive sequence $(C_\ell)_{\ell\in\mathbb{N}}$ with $\ell \cdot C_\ell \to \infty$ and $C_\ell \in \mathcal{O}(\ell^{\beta-1})$ for some $0 < \beta < \frac{1}{\alpha}$. Then for all Borel probability measures $P$ on $X \times Y$ and all $\varepsilon > 0$ it holds

$$\lim_{\ell\to\infty} \mathrm{Pr}^* \left( \{T \in (X \times Y)^\ell \mid \mathcal{R}(f_{T,k,C_\ell}) \le \mathcal{R}^* + \varepsilon\} \right) = 1 \ .$$

The corresponding notation will be introduced in sections 2 and 3. The theorem does not only establish the universal consistency of the multi-class SVM by Crammer and Singer, it also gives precise conditions for how exactly the complexity control parameters needs to be coupled to the training set size in order to obtain universal consistency. Moreover, the rigorous proof of this statement implies that the common belief on the inconsistency of the popular multi-class SVM by Crammer and Singer is wrong. This important learning machine is indeed universally consistent.

## 2 Multi-Class Support Vector Classification

A multi-class classification problem is stated by a training dataset $T = \big((x_1, y_1), \ldots, (x_\ell, y_\ell)\big) \in (X \times Y)^\ell$ with label set $Y$ of size $|Y| = q < \infty$. W.l.o.g., the label space is represented by $Y = \{1, \ldots, q\}$. In contrast to the conceptually simpler binary case we have $q > 2$. The training examples are supposed to be drawn i.i.d. from a probability distribution $P$ on $X \times Y$.

Let $k : X \times X \to \mathbb{R}$ be a positive definite (Mercer) kernel function, and let $\phi : X \to \mathcal{H}$ be a corresponding feature map into a feature Hilbert space $\mathcal{H}$ such that $\langle\phi(x), \phi(x')\rangle = k(x, x')$. We call a function on $X$ induced by $k$ if there exists $w \in \mathcal{H}$ such that $f(x) = \langle w, \phi(x)\rangle$. Let $d_k(x, x') := \|\phi(x) - \phi(x')\|_\mathcal{H} = \sqrt{k(x, x) - 2k(x, x') + k(x', x')}$ denote the metric induced on $X$ by the kernel $k$.

Analog to Steinwart [6] we require that the input space $X$ is a compact subset of $\mathbb{R}^d$, and define the notion of a universal kernel:

**Definition 1.** (Definition 2 in [6]) A continuous positive definite kernel function $k : X \times X \to \mathbb{R}$ on a compact subset $X \subset \mathbb{R}^d$ is called *universal* if the set of induced functions is dense in the space $C^0(X)$ of continuous functions, i.e., for all $g \in C^0(X)$ and all $\varepsilon > 0$ there exists an induced function $f$ with $\|g - f\|_\infty < \varepsilon$.

Intuitively, this property makes sure that the feature space of a kernel is rich enough to achieve consistency for all possible data generating distributions. For a detailed treatment of universal kernels we refer to [6].

An SVM classifier for a $q$-class problem is given in the form of a vector-valued function $f : X \to \mathbb{R}^q$ with component functions $f_u : X \to \mathbb{R}$, $u \in Y$ (sometimes restricted by the so-called sum-to-zero constraint $\sum_{u \in Y} f_u = 0$). Each of its components takes the form $f_u(x) = \langle w_u, \phi(x) \rangle + b_u$ with $w_u \in \mathcal{H}$ and $b_u \in \mathbb{R}$. Then we turn $f$ into a classifier by feeding its result into the 'decision' function

$$\kappa : \mathbb{R}^q \to Y \, ; \qquad (v_1, \ldots, v_q)^T \mapsto \min \left\{ \arg\max\{v_u \mid u \in Y\} \right\} \in Y \ .$$

Here, the arbitrary rule for breaking ties favors the smallest class index.[2] We denote the SVM hypothesis by $h = \kappa \circ f : X \to Y$.

The multi-class SVM variant proposed by Crammer and Singer uses functions without offset terms ($b_u = 0$ for all $u \in Y$). For a given training set $T = \big((x_1, y_1), \ldots, (x_\ell, y_\ell)\big) \in (X \times Y)^\ell$ this machine defines the function $f$, determined by $(w_1, \ldots, w_q) \in \mathcal{H}^q$, as the solution of the quadratic program

$$\text{minimize} \quad \sum_{u \in Y} \langle w_u, w_u \rangle + \frac{C}{\ell} \cdot \sum_{i=1}^{\ell} \xi_i \tag{2}$$

$$\text{s.t.} \quad \langle w_{y_i} - w_u, \phi(x_i) \rangle \geq 1 - \xi_i \qquad \forall\, i \in \{1, \ldots, \ell\},\ u \in Y \setminus \{y_i\} \ .$$

The slack variables in the optimum can be written as

$$\xi_i = \max_{v \in Y \setminus \{y_i\}} \left\{ \big[ 1 - (f_{y_i}(x_i) - f_v(x_i)) \big]_+ \right\} \geq \big[ 1 - \delta_{h(x_i), y_i} - f_{y_i}(x_i) + f_{h(x_i)}(x_i) \big]_+ \ , \tag{3}$$

with the auxiliary function $[t]_+ := \max\{0, t\}$. We denote the function induced by the solution of this problem by $f = f_{T,k,C} = (\langle w_1, \cdot \rangle, \ldots, \langle w_q, \cdot \rangle)^T$.

Let $s(x) := 1 - \max\{P(y|x) \mid y \in Y\}$ denote the noise level, that is, the probability of error of a Bayes optimal classifier. We denote the Bayes risk by $\mathcal{R}^* = \int_X s(x)\,dx$. For a given (measurable) hypothesis $h$ we define its error as $E_h(x) := 1 - P(h(x)|x)$, and its suboptimality w.r.t. Bayes-optimal classification as $\eta_h(x) := E_h(x) - s(x) = \max\{P(y|x) \mid y \in Y\} - P(h(x)|x)$. We have $E_h(x) \geq s(x)$ and thus $\eta_h(x) \geq 0$ up to a zero set.

## 3  The Central Construction

In this section we introduce a number of definitions and constructions preparing the proofs in the later sections. Most of the differences to the binary case are incorporated into these constructions such that the lemmas and theorems proven later on naturally extend to the multi-class case. Let $\Delta := \{p \in \mathbb{R}^q \mid p_u \geq 0 \ \forall\, u \in Y \text{ and } \sum_{u \in Y} p_u = 1\}$ denote the probability simplex over $Y$. We introduce the covering number of the metric space $(X, d_k)$ as

$$\mathcal{N}((X, d_k), \varepsilon) := \min \left\{ n \ \Big| \ \exists\, \{x_1, \ldots, x_n\} \subset X \text{ such that } X \subset \bigcup_{i=1}^{n} B(x_i, \varepsilon) \right\} \ ,$$

with $B(x, \varepsilon) = \{x' \in X \mid d_k(x, x') < \varepsilon\}$ being the open ball of radius $\varepsilon > 0$ around $x \in X$.

Next we construct a partition of a large part of the input space $X$ into suitable subsets. In a first step we partition the probability simplex, then we transfer this partition to the input space, and finally we discard small subsets of negligible impact. The resulting partition has a number of properties of importance for the proofs of diverse lemmas in the next section.

We start by defining $\tau = \varepsilon/(q + 5)$, where $\varepsilon$ is the error bound found in Theorems 1 and 2. Thus, $\tau$ is simply a multiple of $\varepsilon$, which we can think of as an arbitrarily small positive number.

We split the simplex $\Delta$ into a partition of 'classification-aligned' subsets

$$\Delta_y := \kappa^{-1}(\{y\}) = \left\{ p \in \Delta \ \Big| \ p_y > p_u \text{ for } u < y \text{ and } p_y \geq p_u \text{ for } u > y \right\}$$

for $y \in Y$, on which the decision function $\kappa$ decides for class $y$. We define the grid

$$\tilde{\Gamma} = \left\{ [n_1 \tau, (n_1 + 1)\tau) \times \cdots \times [n_q \tau, (n_q + 1)\tau) \subset \mathbb{R}^q \ \Big| \ (n_1, \ldots, n_q)^T \in \mathbb{Z}^q \right\}$$

of half-open cubes. Then we combine both constructions to the partition

$$\Gamma := \bigcup_{y \in Y} \left\{ \tilde{\gamma} \cap \Delta_y \;\middle|\; \tilde{\gamma} \in \tilde{\Gamma} \text{ and } \tilde{\gamma} \cap \Delta_y \neq \emptyset \right\}$$

of $\Delta$ into classification-aligned subsets of side length upper bounded by $\tau$. We have the trivial upper bound $|\Gamma| \leq D := q \cdot (1/\tau + 1)^q$ for the size of the partition. The partition $\Gamma$ will serve as an index set in a number of cases. The first one of these is the partition $X = \bigcup_{\gamma \in \Gamma} X_\gamma$ with $X_\gamma := \{ x \in X \mid P(y|x) \in \gamma \}$.

The compactness of $X$ ensures that the distribution $P$ is regular. Thus, for each $\gamma \in \Gamma$ there exists a compact subset $\tilde{K}_\gamma \subset X_\gamma$ with $P(\tilde{K}_\gamma) \geq (1 - \tau/2) \cdot P(X_\gamma)$. We choose minimal partitions $\tilde{\mathcal{A}}_\gamma$ of each $\tilde{K}_\gamma = \bigcup_{A \in \tilde{\mathcal{A}}_\gamma} A$ such that the diameter of each $A \in \tilde{\mathcal{A}}_\gamma$ is bounded by $\sigma = \tau/(2\sqrt{C})$. All of these sets are summarized in the partition $\tilde{\mathcal{A}} = \bigcup_{\gamma \in \Gamma} \tilde{\mathcal{A}}_\gamma$. Now we drop all $A \in \tilde{\mathcal{A}}_\gamma$ below a certain probabiliy mass, resulting in

$$\mathcal{A}_\gamma := \left\{ A \in \tilde{\mathcal{A}}_\gamma \;\middle|\; P_X(A) \geq \frac{\tau}{2M} \right\} , \tag{4}$$

with $M := D \cdot \mathcal{N}((X, d_k), \sigma)$. We summarize these sets in $K_\gamma = \bigcup_{A \in \mathcal{A}_\gamma} A$ and $\mathcal{A} := \bigcup_{\gamma \in \Gamma} \mathcal{A}_\gamma$. These sets cover nearly all probability mass of $P_X$ in the sense

$$P_X \left( \bigcup_{\gamma \in \Gamma} K_\gamma \right) = P_X \left( \bigcup_{A \in \mathcal{A}} A \right) \geq P_X \left( \bigcup_{A \in \tilde{\mathcal{A}}} A \right) - \tau/2$$

$$= P_X \left( \bigcup_{\gamma \in \Gamma} \tilde{K}_\gamma \right) - \tau/2 \geq P_X \left( \bigcup_{\gamma \in \Gamma} X_\gamma \right) - \tau/2 - \tau/2 = P_X(X) - \tau = 1 - \tau$$

The first estimate makes use of $|\tilde{\mathcal{A}}| \leq M$ and condition (4), while the second inequality follows from the definition of $\tilde{K}_\gamma$.

To simplify notation, we associate a number of quantities with the sets $\gamma \in \Gamma$ and $X_\gamma$. We denote the Bayes-optimal decision by $y(X_\gamma) = y(\gamma) := \kappa(p)$ for any $p \in \gamma$, and for $y \in Y$ we define the lower and upper bounds

$$L_y(X_\gamma) = L_y(\gamma) := \inf \left\{ p_y \;\middle|\; p \in \gamma \right\} \qquad \text{and} \qquad U_y(X_\gamma) = U_y(\gamma) := \sup \left\{ p_y \;\middle|\; p \in \gamma \right\}$$

on the corresponding components in the probability simplex. We canonically extend these definitions to the above defined sets $K_\gamma$, $\tilde{K}_\gamma$, and $A \in \mathcal{A}$, which are all subsets of exactly one of the sets $X_\gamma$, by defining $y(S) := y(\gamma)$ for all non-empty subsets $S \subset X_\gamma$. The resulting construction has the following properties:

(P1) The decision function $\kappa$ is constant on each set $\gamma \in \Gamma$, and thus $h = \kappa \circ f$ is constant on each set $X_\gamma$ as well as on each of their subsets, most importantly on each $A \in \mathcal{A}$.

(P2) For each $y \in Y$, the side length $U_y(\gamma) - L_y(\gamma)$ of each set $\gamma \in \Gamma$ is upper bounded by $\tau$.

(P3) It follows from the construction of $\Gamma$ that for each $y \in Y$ and $\gamma \in \Gamma$ we have either $L_y(\gamma) = 0$ or $L_y(\gamma) \geq \tau$.

(P4) The cardinality of the partition $\Gamma$ is upper bounded by $D = q \cdot (1/\tau + 1)^q$, which depends only on $\tau$ and $q$, but not on $T$, $k$, or $C$.[3]

(P5) The cardinality of the partition $\mathcal{A}$ is upper bounded by $M = D \cdot \mathcal{N}((X, d_k), \tau/(2\sqrt{C}))$, which is finite by Lemma 1.

(P6) The set $\bigcup_{A \in \mathcal{A}} A = \bigcup_{\gamma \in \Gamma} K_\gamma \subset X$ covers a probability mass (w.r.t. $P_X$) of at least $(1 - \tau)$.

(P7) Each $A \in \mathcal{A}$ covers a probability mass (w.r.t. $P_X$) of at least $\frac{\tau}{2M}$.

(P8) Each $A \in \mathcal{A}$ has diameter less than $\sigma = \tau/(2\sqrt{C})$, that is, for $x, x' \in A$ we have $d_k(x, x') < \sigma$.

With properties (P2) and (P6) it is straight-forward to obtain the inequality

$$1 - \sum_{\gamma \in \Gamma} L_{y(\gamma)}(\gamma) \cdot P_X(X_\gamma) - \tau \leq 1 - \sum_{\gamma \in \Gamma} U_{y(\gamma)}(\gamma) \cdot P_X(X_\gamma)$$

$$\leq \mathcal{R}^* \leq 1 - \sum_{\gamma \in \Gamma} L_{y(\gamma)}(\gamma) \cdot P_X(X_\gamma) \leq 1 - \sum_{\gamma \in \Gamma} U_{y(\gamma)}(\gamma) \cdot P_X(X_\gamma) + \tau \qquad (5)$$

for the risk.

Now we are in the position to define the notion of a 'typical' training set. For $\ell \in \mathbb{N}$, $u \in Y$, and $A \in \mathcal{A}$, we define

$$F_\ell^{A,u} := \left\{ ((x_1, y_1), \ldots, (x_\ell, y_\ell)) \in (X \times Y)^\ell \;\middle|\; \right.$$

$$\left. \left| \{ n \in \{1, \ldots, \ell\} \,\middle|\, x_n \in A, y_n = u \} \right| \geq \ell \cdot (1 - \tau) \cdot L_u(A) \cdot P_X(A) \right\} \;.$$

Intuitively, we ask that the number of examples of class $u$ in $A$ does not deviate too much from its expectation, introducing two approximations: The multiplicative factor $(1 - \tau)$, and the lower bound $L_u(A)$ on the conditional probability of class $u$ in $A$. We combine the properties of all these sets in the set $F_\ell := \bigcap_{u \in Y} \bigcap_{A \in \mathcal{A}} F_\ell^{A,u}$ of training sets of size $\ell$, with the same lower bound on the number of training examples in all sets $A \in \mathcal{A}$, and for all classes $u \in Y$.

# 4 Preparations

The proof of our main result follows the proofs of Theorems 1 and 2 in [6] as closely as possible. For the sake of clarity we organize the proof such that all six lemmas in this section directly correspond to Lemmas 1-6 in [6].

**Lemma 1.** (Lemma 1 from [6]) Let $k : X \times X \to \mathbb{R}$ be a universal kernel on a compact subset $X$ or $\mathbb{R}^d$ and $\Phi : X \to \mathcal{H}$ be a feature map of $k$. The $\Phi$ is continuous and

$$d_k(x, x') := \|\Phi(x) - \Phi(x')\|$$

defines a metric on $X$ such that id $: (X, \|\cdot\|) \to (X, d_k)$ is continuous. In particular, $\mathcal{N}((X, d_k), \varepsilon)$ is finite for all $\varepsilon > 0$.

**Lemma 2.** Let $X \subset \mathbb{R}^d$ be compact and let $k : X \times X \to \mathbb{R}$ be a universal kernel. Then, for all $\varepsilon > 0$ and all pairwise disjoint and compact (or empty) subsets $\tilde{K}_u \subset X$, $u \in Y$, there exists an induced function

$$f : X \to \left[ -1/2 \cdot (1 + \varepsilon), 1/2 \cdot (1 + \varepsilon) \right]^q \;; \qquad x \mapsto (\langle w_1^*, x \rangle, \ldots, \langle w_q^*, x \rangle)^T \;,$$

such that

$$f_u(x) \in [1/2, 1/2 \cdot (1 + \varepsilon)] \qquad \text{if } x \in \tilde{K}_u$$

$$f_u(x) \in [-1/2 \cdot (1 + \varepsilon), -1/2] \qquad \text{if } x \in \tilde{K}_v \text{ for some } v \in Y \setminus \{u\}$$

for all $u \in Y$.

*Proof.* This lemma directly corresponds to Lemma 2 in [6], with slightly different cases. Its proof is completely analogous. □

**Lemma 3.** The probability of the training sets $F_\ell$ is lower bounded by

$$P^\ell(F_\ell) \geq 1 - q \cdot M \cdot \exp\left( -\frac{1}{8}(\tau^6/M^2)\ell \right) \;.$$

*Proof.* Let us fix $A \in \mathcal{A}$ and $u \in Y$. In the case $L_u(A) = 0$ we trivially have $P^\ell\big((X \times Y)^\ell \setminus F_\ell^{A,u}\big) = 0$. Otherwise we consider $T = \big((x_1, y_1), \ldots, (x_\ell, y_\ell)\big) \in (X \times Y)^\ell$ and define the binary variables $z_i := \mathbf{1}_{\{A \times \{u\}\}}(x_i, y_i)$, where the indicator function $\mathbf{1}_S(s)$ is one for $s \in S$ and zero

otherwise. This definition allows us to express the cardinality $\left|\{n \in \{1, \ldots, \ell\} \,|\, x_n \in A, y_n = u\}\right| = \sum_{i=1}^{\ell} z_i$ found in the definition of $F_\ell^{A,u}$ in a form suitable for the application of Hoeffding's inequality. The inequality, applied to the variables $z_i$, states

$$P^\ell \left( \sum_{i=1}^{\ell} z_i \leq (1-\tau) \cdot E \cdot \ell \right) \leq \exp\left(-2(\tau E)^2 \ell\right) \quad,$$

where $E := \mathbb{E}[z_i] = \int_{A \times \{u\}} dP(x, y) = \int_A P(u|x) dx \geq L_u(A) \cdot P_X(A) > 0$. Due to $E > 0$ we can use the relation

$$\sum_{i=1}^{\ell} z_i \leq (1-\tau) \cdot E \cdot \ell \;\Rightarrow\; \sum_{i=1}^{\ell} z_i < (1 - \tau/2) \cdot E \cdot \ell$$

in order to obtain Hoeffding's formula for the case of *strict* inequality

$$P^\ell \left( \sum_{i=1}^{\ell} z_i < (1-\tau) \cdot E \cdot \ell \right) \leq \exp\left(-\frac{1}{2}(\tau E)^2 \ell\right) \quad.$$

Combining $E \geq L_u(A) \cdot P_X(A)$ and $\sum_{i=1}^{\ell} z_i < (1-\tau) \cdot L_u(A) \cdot P_X(A) \cdot \ell \Leftrightarrow T \notin F_\ell^{A,u}$ we obtain

$$P^\ell \left( (X \times Y)^\ell \setminus F_\ell^{A,u} \right) = P^\ell \left( \sum_{i=1}^{\ell} z_i < (1-\tau) \cdot L_u(A) \cdot P_X(A) \cdot \ell \right)$$

$$\leq P^\ell \left( \sum_{i=1}^{\ell} z_i < (1-\tau) \cdot E \cdot \ell \right) \leq \exp\left(-\frac{1}{2}(\tau E)^2 \ell\right) \leq \exp\left(-\frac{1}{2}(\tau L_u(A) P_X(A))^2 \ell\right) \quad.$$

Properties (P3) and (P7) ensure $L_u(A) \geq \tau$ and $P_X(A) \geq \tau/(2M)$. Applying these to the previous inequality results in

$$P^\ell \left( (X \times Y)^\ell \setminus F_\ell^{A,u} \right) \leq \exp\left(-\frac{1}{2}(\tau^3/(2M))^2 \ell\right) = \exp\left(-\frac{1}{8}(\tau^6/M^2)\ell\right) \quad,$$

which also holds in the case $L_u(A) = 0$ treated earlier. Finally, we use the union bound

$$1 - P^\ell(F_\ell) = 1 - P^\ell \left( \bigcap_{u \in Y} \bigcap_{A \in \mathcal{A}} F_\ell^{A,u} \right) = P^\ell \left( \bigcup_{u \in Y} \bigcup_{A \in \mathcal{A}} \left( (X \times Y)^\ell \setminus F_\ell^{A,u} \right) \right)$$

$$\leq |Y| \cdot |\mathcal{A}| \cdot \exp\left(-\frac{1}{8}(\tau^6/M^2)\ell\right) \leq q \cdot M \cdot \exp\left(-\frac{1}{8}(\tau^6/M^2)\ell\right)$$

and properties (P4) and (P5) to prove the assertion. $\qquad \square$

**Lemma 4.** The SVM solution $f$ and the hypothesis $h = f \circ \kappa$ fulfill

$$\mathcal{R}(f) \leq \mathcal{R}^* + \int_X \eta_h(x) dx \quad.$$

*Proof.* The lemma follows directly from the definition of $\eta_h$, even with equality. We keep it here because it is the direct counterpart to the (stronger) Lemma 4 in [6]. $\qquad \square$

**Lemma 5.** For all training sets $T \in F_\ell$ the SVM solution given by $(w_1, \ldots, w_q)$ and $(\xi_1, \ldots, \xi_\ell)$ fulfills

$$\sum_{u \in Y} \langle w_u, w_u \rangle + \frac{C}{\ell} \sum_{i=1}^{\ell} \xi_i \;\leq\; \sum_{u \in Y} \langle w_u^*, w_u^* \rangle + C(\mathcal{R}^* + 2\tau) \quad,$$

with $(w_1^*, \ldots, w_q^*)$ as defined in Lemma 2.

*Proof.* The optimality of the SVM solution for the primal problem (2) implies

$$\sum_{u \in Y} \langle w_u, w_u \rangle + \frac{C}{\ell} \sum_{i=1}^{\ell} \xi_i \;\leq\; \sum_{u \in Y} \langle w_u^*, w_u^* \rangle + \frac{C}{\ell} \sum_{i=1}^{\ell} \xi_i^*$$

for any feasible choice of the slack variables $\xi_i^*$. We choose the values of these variables as $\xi_i^* = 1 + \tau$ for $P(y \,|\, x_i) \notin \Delta_{y_i}$ and zero otherwise which corresponds to a feasible solution according to the construction of $w_u^*$ in Lemma 2. Then it remains to show that $\sum_{i=1}^{\ell} \xi_i^* \leq \ell \cdot (\mathcal{R}^* + 2\tau)$. Let $n^+ = \big| \{ i \in \{1, \dots, \ell\} \,|\, P(y \,|\, x_i) \in \Delta_{y_i} \} \big|$ denote the number of training examples correctly classified by the Bayes rule expressed by $\Delta_{y_i}$ (or $\kappa$). Then we have $\sum_{i=1}^{\ell} \xi_i^* = (1 + \tau)(\ell - n^+)$. The definition of $F_\ell$ yields

$$
\begin{aligned}
n^+ &\geq \sum_{u \in Y} \sum_{\substack{A \in \mathcal{A} \\ y(A)=u}} \ell \cdot (1 - \tau) \cdot L_u(\gamma) \cdot P_X(A) = \ell \cdot (1 - \tau) \cdot \sum_{u \in Y} \sum_{\gamma \in \Gamma} \left[ L_u(\gamma) \cdot \sum_{A \in \mathcal{A}_\gamma} P_X(A) \right] \\
&= \ell \cdot (1 - \tau) \cdot \sum_{u \in Y} \sum_{\substack{\gamma \in \Gamma \\ y(\gamma)=u}} \left[ L_u(\gamma) \cdot P_X(K_\gamma) \right] = \ell \cdot (1 - \tau) \cdot \sum_{\gamma \in \Gamma} \left[ L_{y(\gamma)}(\gamma) \cdot P_X(K_\gamma) \right] \\
&\geq \ell \cdot (1 - \tau) \cdot \left( \sum_{\gamma \in \Gamma} \left[ L_{y(\gamma)}(\gamma) \cdot P_X(X_\gamma) \right] - \tau \right) \geq \ell \cdot (1 - \tau) \cdot (1 - \mathcal{R}^*) \;,
\end{aligned}
$$

where the last line is due to inequality (5). We obtain

$$
\begin{aligned}
\sum_{i=1}^{\ell} \xi_i^* &\leq \ell \cdot (1 + \tau) \cdot (1 - (1 - \tau) \cdot (1 - \mathcal{R}^*)) \\
&= \ell \cdot [\mathcal{R}^* + \tau + \tau^2(1 - \mathcal{R}^*)] \leq \ell \cdot [\mathcal{R}^* + \tau + \tau^2] \leq \ell \cdot (\mathcal{R}^* + 2\tau) \;,
\end{aligned}
$$

which proves the claim. $\qquad\square$

**Lemma 6.** For all training sets $T \in F_\ell$ the sum of the slack variables $(\xi_1, \dots, \xi_\ell)$ corresponding to the SVM solution fulfills

$$\sum_{i=1}^{\ell} \xi_i \;\geq\; \ell \cdot (1 - \tau)^2 \cdot \left( \mathcal{R}^* + \int_X \eta_h(x) \, dP_X(x) - q \cdot \tau \right) \;.$$

*Proof.* Problem (2) takes the value $C$ in the feasible solution $w_1 = \dots, w_q = 0$ and $\xi_1 = \dots = \xi_\ell = 1$. Thus, we have $\sum_{u \in Y} \|w_u\|^2 \leq C$ in the optimum, and we deduce $\|w_u\| \leq \sqrt{C}$ for each $u \in Y$. Thus, property (P8) makes sure that $|f_u(x) - f_u(x')| \leq \tau/2$ for all $x, x' \in A$ and $u \in Y$.

The proof works through the following series of inequalities. The details are discussed below.

$$
\sum_{i=1}^{\ell} \xi_i = \sum_{A \in \mathcal{A}} \sum_{u \in Y} \sum_{\substack{x_i \in A \\ y_i = u}} \xi_i
$$

$$
\geq \sum_{A \in \mathcal{A}} \sum_{u \in Y} \sum_{\substack{x_i \in A \\ y_i = u}} \left[ 1 - \delta_{h(x_i),u} + f_{h(x_i)}(x_i) - f_u(x_i) \right]_+
$$

$$
\geq \sum_{A \in \mathcal{A}} \sum_{u \in Y} \sum_{\substack{x_i \in A \\ y_i = u}} \frac{1}{P_X(A)} \cdot \int_A \left[ 1 - \delta_{h(x),u} + f_{h(x)}(x) - f_u(x) - 2 \cdot \frac{\tau}{2} \right]_+ dP_X(x)
$$

$$
\geq \ell \cdot (1 - \tau) \cdot \sum_{A \in \mathcal{A}} \sum_{u \in Y} L_u(A) \cdot \int_A \left[ 1 - \tau - \delta_{h(x),u} + \underbrace{f_{h(x)}(x) - f_u(x)}_{\geq 0} \right]_+ dP_X(x)
$$

$$
\geq \ell \cdot (1 - \tau) \cdot \sum_{A \in \mathcal{A}} \int_A (1 - \tau) \cdot \sum_{u \in Y \setminus \{h(x)\}} L_u(A) \, dP_X(x)
$$

$$
\geq \ell \cdot (1 - \tau)^2 \cdot \sum_{A \in \mathcal{A}} \int_A \left( 1 - q \cdot \tau - L_{h(x)}(A) \right) dP_X(x)
$$

$$
\geq \ell \cdot (1 - \tau)^2 \cdot \sum_{A \in \mathcal{A}} \int_A \left( 1 - q \cdot \tau - 1 + s(x) + \eta_h(x) \right) dP_X(x)
$$

$$
= \ell \cdot (1 - \tau)^2 \cdot \left( \mathcal{R}^* + \int_X \eta_h(x) \, dP_X(x) - q \cdot \tau \right)
$$

The first inequality follows from equation (3). The second inequality is clear from the definition of $F_\ell^{A,u}$ together with $|f_u(x) - f_u(x')| \leq \tau/2$ within each $A \in \mathcal{A}$. For the third inequality we use that the case $u = h(x)$ does not contribute, and the non-negativity of $f_{h(x)}(x) - f_u(x)$. In the next steps we make use of $\sum_{u \in Y} L_u(A) \geq 1 - q \cdot \tau$ and the lower bound $L_{h(x)}(x) \leq P(h(x)|x) = 1 - E_h(x) = 1 - s(x) - \eta_h(x)$, which can be deduced from properties (P1) and (P2). □

## 5 Proof of the Main Result

Just like the lemmas, we organize our theorems analogous to the ones found in [6]. We start with a detailed but technical auxiliaury result.

**Theorem 1.** Let $X \subset \mathbb{R}^d$ be compact, $Y = \{1, \ldots, q\}$, and $k : X \times X \to \mathbb{R}$ a universal kernel. Then, for all Borel probability measures $P$ on $X \times Y$ and all $\varepsilon > 0$ there exists a constant $C^* > 0$ such that for all $C \geq C^*$ and all $\ell \geq 1$ we have

$$
\mathrm{Pr}^* \left( \left\{ T \in (X \times Y)^\ell \mid \mathcal{R}(f_{T,k,C}) \leq \mathcal{R}^* + \varepsilon \right\} \right) \geq 1 - qM \exp\left( -\frac{1}{8} (\tau^6 / M^2) \ell \right) ,
$$

where $\mathrm{Pr}^*$ is the outer probability of $P^\ell$, $f_{T,k,C}$ is the solution of problem (2), $M = q \cdot (1/\tau + 1)^q \cdot \mathcal{N}((X, d_k), \tau/(2\sqrt{C}))$, and $\tau = \varepsilon/(q + 5)$.

*Proof.* According to Lemma 3 it is sufficient to show $\mathcal{R}(f_{T,k,C}) \leq \mathcal{R}^* + \varepsilon$ for all $T \in F_\ell$. Lemma 4 provides the estimate $\mathcal{R}(f) \leq \mathcal{R}^* + \int_X \eta_h(x) \, dP_X(x)$, such that it remains to show that $\int_X \eta_h(x) \, dP_X(x) \leq \varepsilon$ for $T \in F_\ell$. Consider $w_u^*$ as defined in Lemma 2, then we combine Lemmas 5 and 6 to

$$
(1 - \tau)^2 \cdot \left( \underbrace{\mathcal{R}^* + \int_X \eta_h(x) \, dP_X(x) - q \cdot \tau}_{\leq 1} \right) \leq \frac{1}{C} \left( \sum_{u \in Y} \|w_u^*\|^2 - \underbrace{\sum_{u \in Y} \|w_u\|^2}_{\geq 0} \right) + (\mathcal{R}^* + 2\tau) .
$$

Using $a-\tau \leq (1-\tau)\cdot a$ for any $a \in [0,1]$, we derive $\int_X \eta_h(x)\, dP_X(x) \leq \frac{1}{C}\sum_{u \in Y} \|w_u^*\|^2 + (q+4)\cdot \tau$. With the choice $C^* = \frac{1}{\tau}\cdot\sum_{u \in Y}\|w_u^*\|^2$ and the condition $C \geq C^*$ we obtain $\int_X \eta_h(x)\, dP_X(x) \leq (q+5)\cdot \tau = \varepsilon$. $\qquad\square$

*Proof of Theorem 2.* Up to constants, this short proof coincides with the proof of Theorem 2 in [6]. Because of the importance of the statement and the brevity of the proof we repeat it here:

Since $\ell \cdot C_\ell \to \infty$ there exists an integer $\ell_0$ such that $\ell \cdot C_\ell \geq C^*$ for all $\ell \geq \ell_0$. Thus for $\ell \geq \ell_0$ Theorem 1 yields

$$\mathrm{Pr}^*\left(\left\{T \in (X \times Y)^\ell \;\middle|\; \mathcal{R}(f_{T,k,C_\ell}) \leq \mathcal{R}^* + \varepsilon\right\}\right) \geq 1 - qM_\ell \exp\left(-\frac{1}{8}(\tau^6/M_\ell^2)\ell\right) \;,$$

where $M_\ell = D \cdot \mathcal{N}((X,d_k), \tau/(2\sqrt{C_\ell}))$. Moreover, by the assumption on the covering numbers of $(X, d_k)$ we obtain $M_\ell^2 \in \mathcal{O}((\ell \cdot C_\ell)^2)$ and thus $\ell \cdot M_\ell^{-2} \to \infty$. $\qquad\square$

# 6 Conclusion

We have proven the universal consistency of the popular multi-class SVM by Crammer and Singer. This result disproves the common belief that this machine is in general inconsistent. The proof itself can be understood as an extension of Steinwart's universal consistency result for binary SVMs. Just like there are different extensions of the binary SVM to multi-class classification in the literature, we strongly believe that our proof can be further generalized to cover other multi-class machines, such as the one proposed by Weston and Watkins, which is a possible direction for future research.

## Footnotes

[1]For $f, g : \mathbb{R}^+ \to \mathbb{R}^+$ we define $f(x) \in \mathcal{O}(g(x))$ iff $\exists c, x_0 > 0$ such that $f(x) \le c \cdot g(x) \ \forall x > x_0$.

[2] Note that any other deterministic rule for breaking ties can be realized by permuting the class indices.

[3]A tight bound would be in $\mathcal{O}(\tau^{1-q})$.

# References

[1] C. Cortes and V. Vapnik. Support-Vector Networks. *Machine Learning*, 20(3):273–297, 1995.

[2] K. Crammer and Y. Singer. On the algorithmic implementation of multiclass kernel-based vector machines. *Journal of Machine Learning Research*, 2:265–292, 2002.

[3] S. Hill and A. Doucet. A Framework for Kernel-Based Multi-Category Classification. *Journal of Artificial Intelligence Research*, 30:525–564, 2007.

[4] Y. Lee, Y. Lin, and G. Wahba. Multicategory Support Vector Machines: Theory and Application to the Classification of Microarray Data and Satellite Radiance Data. *Journal of the American Statistical Association*, 99(465):67–82, 2004.

[5] Y. Liu. Fisher Consistency of Multicategory Support Vector Machines. *Journal of Machine Learning Research*, 2:291–298, 2007.

[6] I. Steinwart. Support Vector Machines are Universally Consistent. *J. Complexity*, 18(3):768–791, 2002.

[7] A. Tewari and P. L. Bartlett. On the Consistency of Multiclass Classification Methods. *Journal of Machine Learning Research*, 8:1007–1025, 2007.

[8] V. Vapnik. *Statistical Learning Theory*. Wiley, New-York, 1998.

[9] J. Weston and C. Watkins. Support vector machines for multi-class pattern recognition. In M. Verleysen, editor, *Proceedings of the Seventh European Symposium On Artificial Neural Networks (ESANN)*, pages 219–224, 1999.

